# OPTIMAL NEURAL SPIKE CLASSIFICATION

Amir F. Atiya(*) and James M. Bower(**)
(*) Dept. of Electrical Engineering
(**) Division of Biology
California Institute of Technology
Ca 91125

## Abstract

Being able to record the electrical activities of a number of neurons simultaneously is likely to be important in the study of the functional organization of networks of real neurons. Using one extracellular microelectrode to record from several neurons is one approach to studying the response properties of sets of adjacent and therefore likely related neurons. However, to do this, it is necessary to correctly classify the signals generated by these different neurons. This paper considers this problem of classifying the signals in such an extracellular recording, based upon their shapes, and specifically considers the classification of signals in the case when spikes overlap temporally.

## Introduction

How single neurons in a network of neurons interact when processing information is likely to be a fundamental question central to understanding how real neural networks compute. In the mammalian nervous system we know that spatially adjacent neurons are, in general, more likely to interact, as well as receive common inputs. Thus neurobiologists are interested in devising techniques that allow adjacent groups of neurons to be sampled simultaneously. Unfortunately, the small scale of real neural networks makes inserting one recording electrode per cell impractical. Therefore, one is forced to use single electrodes designed to sample neural signals evoked by several cells at once. While this approach provides the multi-neuron recordings being sought, it also presents a rather serious waveform classification problem because the actual temporal sequence of action potentials in each individual neuron must be deciphered. This paper describes a method for classifying the activities of several individual neurons recorded simultaneously using a single electrode.

## Description of the Problem

Over the last two decades considerable attention[1-8] has been devoted to the problem of classification of action potentials in multi-neuron recordings. These action potentials (also referred to as "spikes") are the extracellularly recorded signal produced by a single neuron when it is passing information to other neurons (Fig. 1). Fortunately, spikes recorded from the same cell are more or less similar in shape, while spikes coming from different neurons usually have somewhat different shapes, depending on the neuron type, electrode characteristics, the distance between the electrode and the neuron, and the intervening medium. Fig. 1 illustrates some representative variations in spike shapes. It is our objective to detect and classify different spikes based on their shapes. However, relying entirely on the shape of the spikes presents difficulties. For example spikes from different neurons can overlap temporally producing novel waveforms (see Fig. 2 for an example of an overlap). To deal with these overlaps, one has first to detect the occurrence of an overlap, and then estimate the constituent spikes. Unfortunately, only a few of the available spike separation algorithms consider these events, even though they are potentially very important in understanding neural networks. Those few tend to rely

on heuristic rules and subtractive methods to resolve overlap cases. No currently published method we are aware of attempts to use knowledge of the likelihood of overlap events for detecting them, which is at the basis of the method we will describe.

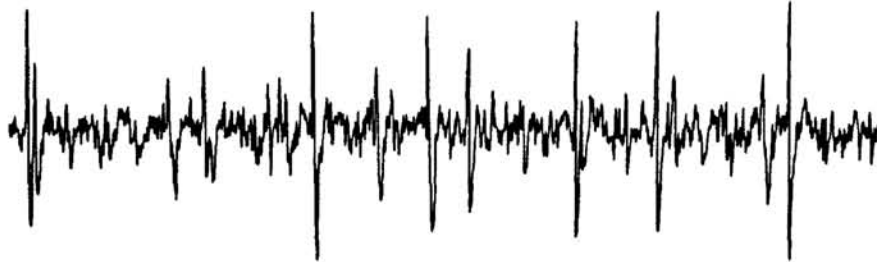

Fig. 1
An example of a multi-neuron recording

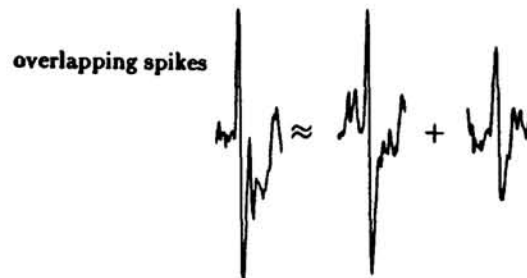

Fig. 2
An example of a temporal overlap of action potentials

**General Approach**

The first step in classifying neural waveforms is obviously to identify the typical spike shapes occurring in a particular recording. To do this we have applied a learning algorithm on the beginning portion of the recording, which in an unsupervised fashion (i.e. without the intervention of a human operator) estimates the shapes. After the learning stage we have the classification stage, which is applied on the remaining portion of the recording. A new classification method is proposed, which gives minimum probability of error, even in case of the occurrence of overlapping spikes. Both the learning and the classification algorithms require a preprocessing step to detect the position of the spike candidate in the data record.

*Detection:* For the first task of detection most researchers use a simple level detecting algorithm, that signals a spike when recorded voltage levels cross a certain voltage threshold. However, variations in recording position due to natural brain movements during recording (e.g. respiration) can cause changes in relative height of the positive to the negative peak. Thus, a level detector (using either a positive or a negative threshold) can miss some spikes. Alternatively, we have chosen to detect an event by sliding a window of fixed length until a time when the peak to peak value within the window exceeds a certain threshold.

*Learning:* Learning is performed on the beginning portion of the sampled data using the Isodata clustering algorithm[9]. The task is to estimate the number of neurons $n$ whose spikes are represented in the waveform and learn the different shapes of the spikes of the various neurons. For that purpose we apply the clustering algorithm choosing only one feature

from the spike, the peak to peak value which we have found to be quite an effective feature. Note that using the peak to peak value in the learning stage does not necessitate using it for classification (one might need additional or different features, especially for tackling the case of spike overlap).

*The Optimal Classification Rule:* Once we have identified the number of different events present, the classification stage is concerned with estimating the identities of the spikes in the recording, based on the typical spike shapes obtained in the learning stage. In our classification scheme we make use of the information given by the shape of the detected spike as well as the firing rates of the different neurons. Although the shape plays in general the most important role in the classification, the rates become a more significant factor when dealing with overlapping events. This is because in general overlap is considerably less frequent than single spikes. The shape information is given by a set of features extracted from the waveform. Let x be the feature vector of the detected spike (e.g. the samples of the spike waveform). Let $N_1, ..., N_n$ represent the different neurons. The detection algorithm tells us only that at least one spike occurred in the narrow interval $(t - T_1, t + T_2)$ (= say $I$) where $t$ is the instant of the peak of the detected spike, $T_1$ and $T_2$ are constants chosen subjectively according to the smallest possible time separation between two consecutive spikes, identifiable as two separate (nonoverlapping) spikes. By definition, if more than one spike occurs in the interval $I$, then we have an overlap. As a matter of convention, the instant of the occurrence of a spike is taken to be that of the spike peak. For simplicity, we will consider the case of two possibly overlapping spikes, though the method can be extended easily to more. The classification rule which results in minimum probability of error is the one which chooses the neuron (or pair of neurons in case of overlap) which has the maximum likelihood. We have therefore to compare the $P_i$'s and the $P_{lj}$'s, defined as

$$P_i = P(N_i \text{ fired in } I | \mathbf{x}, A), \qquad i = 1, ..., n$$

$$P_{lj} = P(N_l \text{ and } N_j \text{ fired in } I | \mathbf{x}, A), \qquad l, j = 1, ..., n, \quad j < l$$

where $A$ represents the event that one or two spikes occurred in the interval $I$. In other words $P_i$ the probability that what has been detected is a single spike from neuron $i$, whereas $P_{lj}$ is the probability that we have two overlapping spikes from neurons $l$ and $j$ (note that spikes from the same neuron never overlap). Henceforth we will use $f$ to denote probability density. For the purpose of abbreviation let $B_i(t)$ mean "neuron $N_i$ fired at $t$". The classification problem can be reduced to comparing the following likelihood functions:

$$L_i = f(B_i(t)) \int_{t-T_1}^{t+T_2} f(\mathbf{x} | B_i(t_1)) dt_1, \qquad i = 1, ..., n \qquad (1a)$$

$$L_{lj} = f(B_j(t)) f(B_l(t)) \int_{t-T_1}^{t+T_2} \int_{t-T_1}^{t+T_2} f(\mathbf{x} | B_l(t_1), B_j(t_2)) dt_1 dt_2, \qquad l, j = 1, ..., n, \quad j < l \qquad (1b)$$

(for a derivation refer to Appendix). Let $f_i$ be the density of the inter-spike interval and $\tau_i$ be the most recent firing instant of neuron $N_i$. If we are given the fact that neuron $N_i$ has been idle for at least a period of duration $t - \tau_i$, we get

$$f(B_i(t)) = \frac{f_i(t - \tau_i)}{\int_{t-\tau_i}^{\infty} f_i(\tau) d\tau}. \qquad (2)$$

A disadvantage of using (2) is that the available $f_i$'s and $\tau_i$'s are only estimates, which depend on the previous classification results, Further, for reliable estimation of the densities $f_i$, one needs a large number of spikes and therefore a long learning period since we are estimating a

whole function. Therefore, we have not used this form, but instead have used the following two schemes. In the first one, we ignore the knowledge about the previous firing pattern except for the estimated firing rates $\lambda_1, ..., \lambda_n$ of the different neurons $N_1, ..., N_n$ respectively. Then the probability of a spike coming from neuron $N_i$ in an interval of duration $dt$ is simply $\lambda_i dt$. Hence

$$f(B_i(t)) = \lambda_i. \qquad (3)$$

In the second scheme we do not use any previous knowledge except for the total firing rate (of all neurons), say $\alpha$. Then

$$f(B_i(t)) = \frac{\alpha}{n}. \qquad (4)$$

Although the second scheme does not use as much of the information about the firing pattern as the first scheme does, it has the advantage of obtaining and using a more reliable estimate of the firing rate, because in general the overall firing rate changes less with time than the individual rates and because the estimate of $\alpha$ does not depend on previous classification results. However, it is useful mostly when the firing rates of the different neurons do not vary much, otherwise the firt scheme is preferred.

In real recording situations, sometimes one encounters voltage signals which are much different than any of the previously learned typical spike shapes or their pairwise overlaps. This can happen for example due to a falsely detected noise event, a spike from a class not encountered in the learning stage, or to the overlap of three or more spikes. To cope with these cases we use the reject option. This means that we refuse to classify the detected spike because of the unlikeliness of the assumed event $A$. The reject option is therefore employed whenever $P(A|\mathbf{x})$ is smaller than a certain threshold. We know that

$$P(A|\mathbf{x}) = f(A,\mathbf{x})/[f(A,\mathbf{x}) + f(A^c,\mathbf{x})]$$

where $A^c$ is the complement of the event $A$. The density $f(A^c,\mathbf{x})$ can be approximated as uniform (over the possible values of $\mathbf{x}$) because a large variety of cases are covered by the event $A^c$. It follows that one can just compare $f(A,\mathbf{x})$ to a threshold. Hence the decision strategy becomes finally: Reject if the sum of the likelihood functions is less than a threshold. Otherwise choose the neuron (or pair of neurons) corresponding to the largest likelihood functions. Note that the sum of the likelihood functions equals $f(A,\mathbf{x})$ (refer to Appendix).

Now, let us evaluate the integrals in (1). Overlapping spikes are assumed to add linearly. Since we intend to handle the overlap case, we have to use a set of features $x_m$ which obeys the following. Given the features of two of the waveforms, then one can compute those of their overlap. A good such candidate is the set of the samples of the spike (or possibly also just part of the samples). The added noise, partly thermal noise from the electrode and partly due to firings from distant neurons, can usually be approximated as white Gaussian. Let the variance be $\sigma^2$. The integrals in the likelihood functions can be approximated as summations (note in fact that we have samples available, not a continuous waveform). Let $\mathbf{y}^i$ represent the typical feature vector (template) associated with neuron $N_i$, with the $m^{th}$ component being $y_m^i$. Then

$$f(\mathbf{x}|B_i(k_1)) = \frac{1}{(2\pi)^{M/2}\sigma^M} exp\Big[-\frac{1}{2\sigma^2}\sum_{m=1}^{M}(x_m - y_{m-k_1}^i)^2\Big]$$

$$f(\mathbf{x}|B_l(k_1), B_j(k_2)) = \frac{1}{(2\pi)^{M/2}\sigma^M} exp\Big[-\frac{1}{2\sigma^2}\sum_{m=1}^{M}(x_m - y_{m-k_1}^l - y_{m-k_2}^j)^2\Big]$$

where $x_m$ is the $m^{th}$ component of $\mathbf{x}$, and $M$ is the dimension of $\mathbf{x}$. This leads to the following likelihood functions

$$L'_i = f(B_i(k)) \sum_{k_1=-M_1}^{M_2} exp\big[-\frac{1}{2\sigma^2} \sum_{m=1}^{M} (x_m - y^i_{m-k_1})^2\big]$$

$$L'_{lj} = f(B_l(k))f(B_j(k)) \sum_{k_1=-M_1}^{M_2} \sum_{k_2=-M_1}^{M_2} exp\big[-\frac{1}{2\sigma^2} \sum_{m=1}^{M} (x_m - y^l_{m-k_1} - y^j_{m-k_2})^2\big]$$

where $k$ is the spike instant, and the interval from $-M_1$ to $M_2$ corresponds to the interval $I$ defined at the beginning of the Section.

## Implementation

The techniques we have just described were tested in the following way. For the first experiment we identified two spike classes in a recording from the rat cerebellum. A signal is created, composed of a number of spikes from the two classes at random instants, plus noise. To make the situation as realistic as possible, the added noise is taken from idle periods (i.e. non-spiking) of a real recording. The reason for using such an artificially generated signal is to be able to know the class identities of the spikes, in order to test our approach quantitatively. We implement the detection and classification techniques on the obtained signal, with various values of noise amplitude. In our case the ratio of the peak to peak values of the templates turns out to be 1.375. Also, the spike rate of one of the clases is twice that of the other class. Fig.3a shows the results with applying the first scheme (i.e. using Eq. 3). The overall percentage correct classification for all spikes (solid curve) and the percentage correct classification for overlapping spikes (dashed curve) are plotted versus the standard deviation of the noise $\sigma$ normalized with respect to the peak $h$ of the large template. Notice that the overall classification accuracy is near 100% for $\sigma/h$ less than 0.15, which is actually the range of noise amplitudes we mostly encountered in our work with real recordings. Observe also the good results for classifying overlapping events. We have applied also the second scheme (i.e. using Eq. 4) and obtained similar results. We wish to mention that the thresholds for detection and for the reject option are set up so as to obtain no more than 3% falsely detected spikes.

A similar experiment is performed with three waveforms (three classes), where two of the waveforms are the same as those used in the first experiment. The third is the average of the first two. All the three neurons have the same spike rate (i.e. $\lambda_1 = \lambda_2 = \lambda_3$). Hence both classification schemes are equivalent in this case. Fig. 3b shows the overall as well as the sub-category of overlap classification results. One observes that the results are worse than those for the two-class case. This is because the spacings between the templates are in general smaller. Notice also that the accuracy in resolving overlapping events is now tangibly less than the overall accuracy. However, one can say that the results are acceptable in the range of $\sigma/h$ less than 0.1. The following experiment is also performed using the same data. We would like to investigate the importance of the information given by the (overall) firing rate on the problem of classifying overlapping events. In our method the summation in the likelihood functions for single spikes is multiplied by $\alpha/n$, while that for overlapping spikes is multiplied by $(\alpha/n)^2$. Usually $\alpha/n$ is considerably less than one. Hence we have a factor which gives less weight for overlapping events. Now, consider the case of ignoring completely the information given by the firing rate and relying solely on shape information. We assume that overlapping spikes from any two given classes represent "new" class of waveforms and that each of these overlap classes has the same rate as that of a single-spike class. In that case we can obtain expressions for the likelihood functions as consisting just the summations, i.e. free of the rate

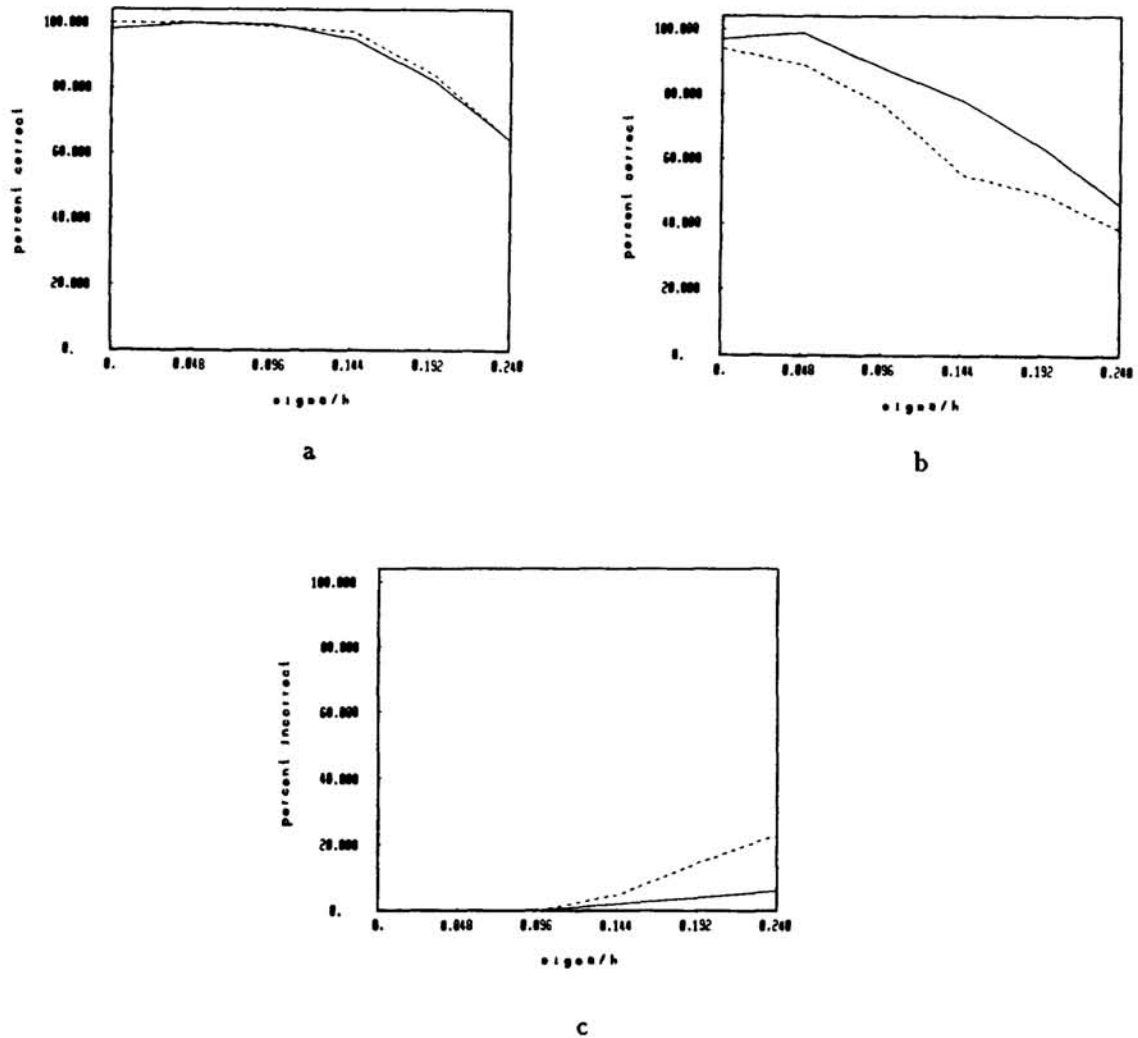

Fig. 3
a) Overall (solid curve) and overlap (dashed curve)
classification accuracy for a two class case
b) Overall (solid curve) and overlap (dashed curve)
classification accuracy for a three class case
c)Percent of incorrect classification of single spikes as overlap
solid curve: scheme utilzing the spike rate
dashed curve: scheme not utilizing the spike rate

factor $\alpha/n$ (refer to Appendix). An experiment is performed using that scheme (on the same three class data). One observes that the method classifies a number of single spikes wrongly as overlaps, much more than our original scheme does (see Fig. 3c), especially for the large noise case. On the other hand, the number of overlaps which are classified wrongly as single spikes is near zero for both schemes.

Finally, in the last experiment the techniques are implemented on real recordings from the rat cerebellum. The recorded signal is band-pass-filtered in the frequency range 300 Hz - 10 KHz, then sampled with a rate of 20KHz. For classification, we take 20 samples per spike as features. Fig. 4 shows the results of the proposed method, using the first scheme (Eq. 3). The number of neurons whose spikes are represented in the waveform is estimated to be four. The

detection threshold is set up so that spikes which are too small are disregarded, because they come from several neurons far away from the electrode and are hard to distinguish. Notice the overlap of classes 1 and 2, which was detected. We used the second scheme also on the same portion and it gave similar results as those of the first scheme (only one of the spikes is classified differently). Overall, the discrepancies between classifications done by the proposed method and an experienced human observer were found to be small.

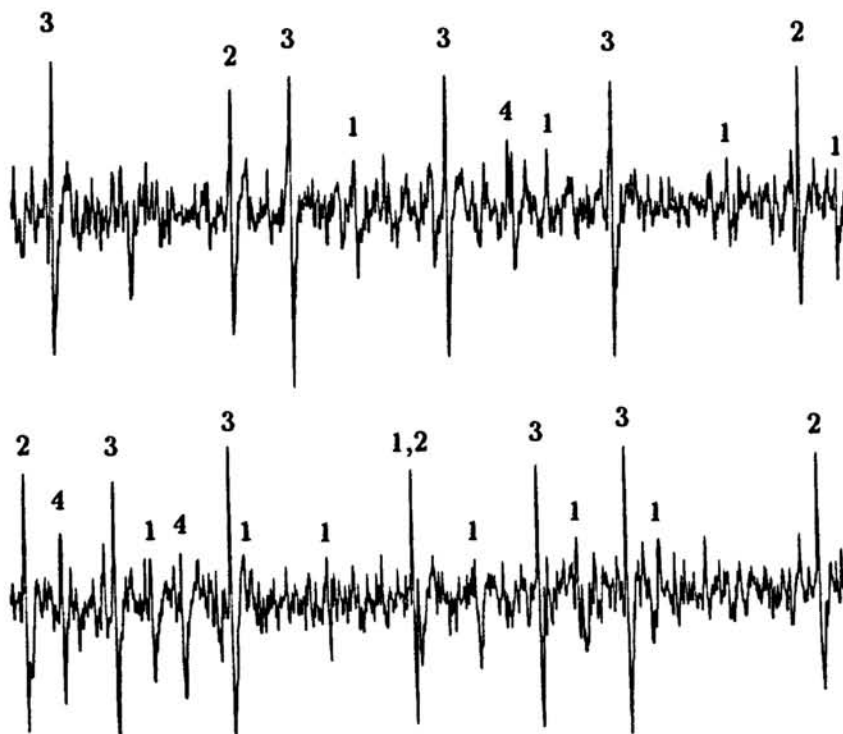

Fig. 4
Classification results for a recording from the rat cerebellum

## Conclusion

Many researchers have considered the problem of spike classification in multi-neuron recordings, but only few have tackled the case of spike overlap, which could occur frequently, particularly if the group of neurons under study is stimulated. In this work we propose a method for spike classification, which can also aid in detecting and classifying overlapping spikes. By taking into account the statistical properties of the discharges of the neurons sampled, this method minimizes the probability of classification error. The application of the method to artificial as well as real recordings confirm its effectiveness.

## Appendix

Consider first $P_{lj}$. We can write

$$P_{lj} = \int_{t-T_1}^{t+T_2} \int_{t-T_1}^{t+T_2} f(B_l(t_1), B_j(t_2)|\mathbf{x}, A) dt_1 dt_2.$$

We can also obtain

$$P_{lj} = \int_{t-T_1}^{t+T_2} \int_{t-T_1}^{t+T_2} \frac{f(\mathbf{x}, A | B_l(t_1), B_j(t_2))}{f(\mathbf{x}, A)} f(B_l(t_1), B_j(t_2)) dt_1 dt_2.$$

Now, consider the two events $B_l(t_1)$ and $B_j(t_2)$. In the absense of any information about their dependence, we assume that they are independent. We get

$$f(B_l(t_1), B_j(t_2)) = f(B_l(t_1)) f(B_j(t_2)).$$

Within the interval $I$, both $f(B_l(t_1))$ and $f(B_j(t_2))$ hardly vary because the duration of $I$ is very small compared to a typical inter-spike interval. Therefore we get the following approximation:

$$f(B_l(t_1)) \approx f(B_l(t))$$
$$f(B_j(t_2)) \approx f(B_j(t)).$$

The expression for $P_{lj}$ becomes

$$P_{lj} \approx \frac{f(B_l(t)) f(B_j(t))}{f(\mathbf{x}, A)} \int_{t-T_1}^{t+T_2} \int_{t-T_1}^{t-T_2} f(\mathbf{x} | B_l(t_1), B_j(t_2)) dt_1 dt_2.$$

Notice that the term $A$ was omitted from the argument of the density inside the integral, because the occurrence of two spikes at $t_1$ and $t_2 \epsilon I$ implies the occurrence of $A$. A similar derivation for $P_i$ results in

$$P_i \approx \frac{f(B_i(t))}{f(\mathbf{x}, A)} \int_{t-T_1}^{t+T_2} f(\mathbf{x} | B_i(t_1)) dt_1.$$

The term $f(\mathbf{x}, A)$ is common to all the $P_{lj}$'s and the $P_i$'s. Hence one can simply compare the following likelihood functions:

$$L_i = f(B_i(t)) \int_{t-T_1}^{t+T_2} f(\mathbf{x} | B_i(t_1)) dt_1$$

$$L_{lj} = f(B_l(t)) f(B_j(t)) \int_{t-T_1}^{t+T_2} \int_{t-T_1}^{t+T_2} f(\mathbf{x} | B_l(t_1), B_j(t_2)) dt_1 dt_2.$$

### Acknowledgement

Our thanks to Dr. Yaser Abu-Mostafa for his assistance with this work. This project was supported by the Caltech Program of Advanced Technology (sponsored by Aerojet, GM, GTE, and TRW), and the Joseph Drown Foundation.

### References

[1] M. Abeles and M. Goldstein, *Proc. IEEE*, 65, pp.762-773, 1977.

[2] G. Dinning and A. Sanderson, *IEEE Trans. Bio − Med. Eng.*, BME-28, pp. 804-812, 1981.

[3] E. D'Hollander and G. Orban, *IEEE Trans. Bio-Med. Eng.*, BME-26, pp. 279-284, 1979.

[4] D. Mishelevich, *IEEE Trans. Bio-Med. Eng.*, BME-17, pp. 147-150, 1970.

[5] V. Prochazka and H. Kornhuber, *Electroenceph. clin. Neurophysiol.*, 32, pp. 91-93, 1973.

[6] W. Roberts, *Biol. Cybernet.*, 35, pp. 73-80, 1979.

[7] W. Roberts and D. Hartline, *Brain Res.*, 94, pp. 141-149, 1975.

[8] E. Schmidt, *J. Neurosci. Methods*, 12, pp. 95-111, 1984.

[9] R. Duda and P. Hart, *Pattern Classification and Scene Analysis*, John Wiley, 1973.
